# NEURAL APPROACH FOR TV IMAGE COMPRESSION USING A HOPFIELD TYPE NETWORK

Martine NAILLON     Jean-Bernard THEETEN
Laboratoire d'Electronique et de Physique Appliquée *
3 Avenue DESCARTES, BP 15
94451 LIMEIL BREVANNES  Cedex  FRANCE.

## ABSTRACT

A self-organizing Hopfield network has been developed in the context of Vector Quantiza--tion, aiming at compression of television images. The metastable states of the spin glass-like network are used as an extra storage resource using the Minimal Overlap learning rule (Krauth and Mezard 1987) to optimize the organization of the attractors. The self-organizing scheme that we have devised results in the generation of an adaptive codebook for any given TV image.

## INTRODUCTION

The ability of an Hopfield network (Little,1974; Hopfield,1982,1986; Amit. and al., 1987; Personnaz and al. 1985; Hertz, 1988) to behave as an associative memory usually assumes a priori knowledge of the patterns to be stored. As in many applications they are unknown, the aim of this work is to develop a network capable to learn how to select its attractors. TV image compression using Vector Quantization (V.Q.)(Gray, 1984), a key issue for HDTV transmission, is a typical case, since the non neural algorithms which generate the list of codes (the codebook) are suboptimal. As an alternative to the promising neural compression techniques (Jackel et al., 1987; Kohonen, 1988; Grossberg, 1987; Cottrel et al., 1987) our idea is to use the metastability in a spin glass-like net as an additional storage resource and to derive after a "classical" clustering algorithm a self-organizing sheme for generating adaptively the codebook. We present the illustrative case of 2D-vectors.

* LEP : A member of the Philips Research Organization.

## NON NEURAL APPROACH

In V.Q., the image is divided into blocks, named vectors, of N pixels (typically 4 x 4 pixels). Given the codebook, each vector is coded by associating it with the nearest element of the list (Nearest Neighbour Classifier) (figure 1).

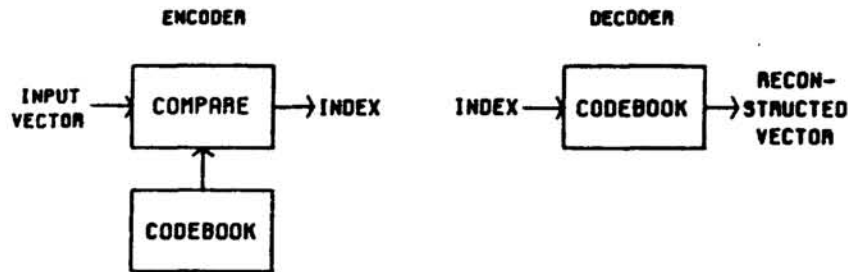

Figure 1 : Basic scheme of a vector quantizer.

For designing an optimal codebook, a clustering algorithm is applied to a training set of vectors (figure 2), the criterium of optimality being a distorsion measure between the training set and the codebook. The algorithm is actually suboptimal, especially for non connex training set, as it is based on an iterative computation of centers of gravity which tends to overcode the dense regions of points whereas the light ones are undercoded (figure 2).

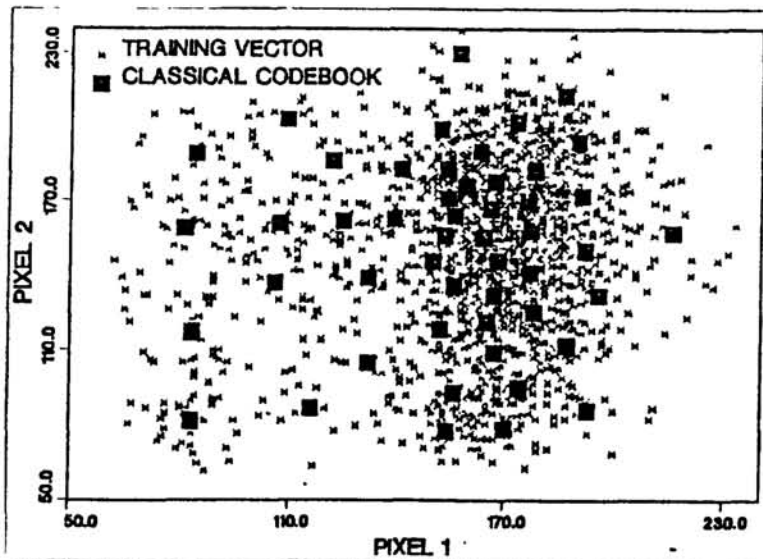

Figure 2 : Training set of two pixels vectors and the associated codebook computed by a non neural clustering algorithm : overcoding of the dense regions (pixel 1 148) and subcoding of the light ones.

## NEURAL APPROACH

In a Hopfield neural network, the code vectors are the attractors of the net and the neural dynamics (resolution phase) is substituted to the nearest neighbourg classification.

When patterns - referred to as "prototypes" and named here "explicit memory" - are prescribed in a spin glass-like net, other attractors - referred to as "metastable states" - are induced in the net (Sherrington and Kirkpatrick, 1975; Toulouse, 1977; Hopfield, 1982; Mezard and al., 1984). We consider those induced attractors as additional memory named here "implicit memory" which can be used by the network to code the previously mentioned light regions of points. This provides a higher flexibility to the net during the self-organization process, as it can choose in a large basis of explicit and implicit attractors the ones which will optimize the coding task.

NEURAL NOTATION

A vector of 2 pixels with 8 bits per pel is a vector of 2 dimensions in an Euclidean space where each dimension corresponds to 256 grey levels. To preserve the Euclidean distance, we use the well-known themometric notation : 256 neurons for 256 levels per dimension, the number of neurons set to one, with a regular ordering, giving the pixel luminance, e.g. 2 = 1 1-1-1-1-1... For vectors of dimension 2, 512 neurons will be used, e.g. v=(2,3)= (1 1-1-1......-1,1 1 1-1-1...,-1)

INDUCTION PROCESS

The induced implicit memory depends on the prescription rule. We have compared the Projection rule (Personnaz and al., 1985) and the Minimal Overlap rule (Krauth and Mezard, 1987).

The metastable states are detected by relaxing any point of the training set of the figure 2, to its corresponding prescribe or induced attractor marked in figure 3 with a small diamond.

For the two rules, the induction process is rather deterministic, generating an orthogonal mesh : if two prototypes $(P_{11},P_{12})$ and $(P_{21},P_{22})$ are prescribed, a metastable state is induced at the cross-points, namely $(P_{11},P_{22})$ and $(P_{21},P_{12})$ (figure 3).

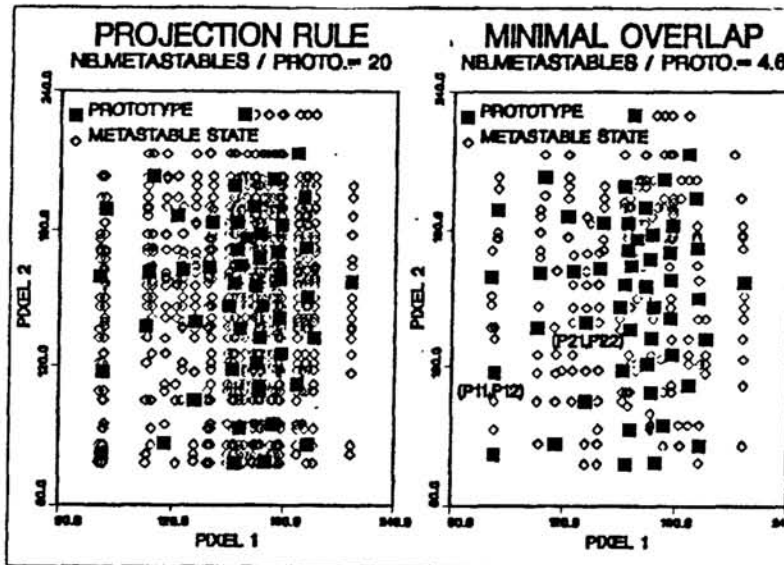

Figure 3 : Comparaison of the induction process for 2 prescription rules. The prescribed states are the full squares, the induced states the open diamonds.

What differs between the two rules is the number of induced attractors. For 50 prototypes and a training set of 2000 2d-vectors, the projection rule induces about 1000 metastable states (ratio 1000/50 = 20) whereas Min Over induces only 234 (ratio 4.6). This is due to the different stability of the prescribed and the induced states in the case of Min Over (Naillon and Theeten, to be published).

GENERALIZED ATTRACTORS
Some attractors are induced out of the image space (Figure 4) as the 512 neurons space has $2^{512}$ configurations to be compared with the $(2^8)^2 = 2^{16}$ image configurations.

We extend the image space by defining a "generalized attractor" as the class of patterns having the same number of neurons set to one for each pixel, whatever their ordering. Such a notation corresponds to a random thermometric neural representation. The simulation has shown that the generalized attractors correspond to acceptable states (Figure 4) i.e. they are located at the place when one would like to obtain a normal attractor.

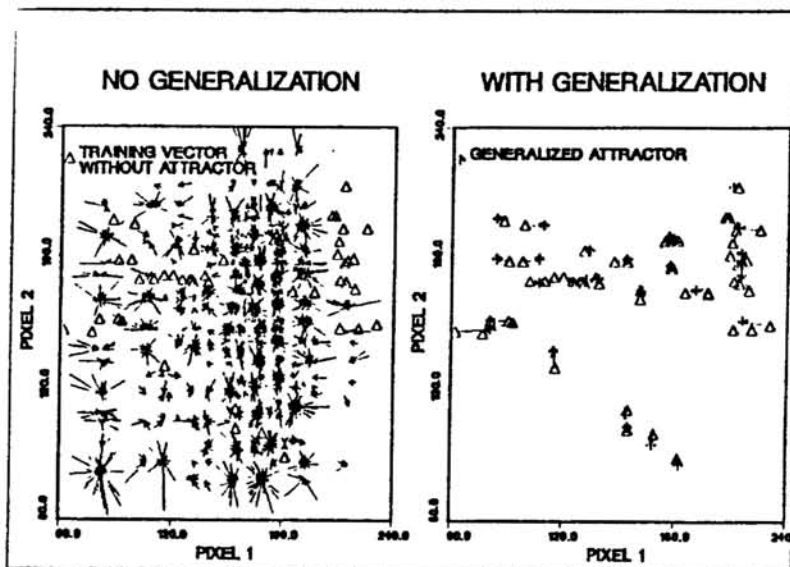

Figure 4 : The induced bassins of attractions are represented with arrows. In the left plot, some training vectors have no attractor in the image space. After generalization (randon thermometric notation), the right plot shows their corresponding attractors.

## ADAPTIVE NEURAL CODEBOOK LEARNING

An iterative self-organizing process has been developed to optimize the codebook. For a given TV image, the codebook is defined, at each step of the process, as the set of prescribed and induced attractors, selected by the training set of vectors. The self-organizing scheme is controlled by a cost function, the distorsion measure between the training set and the codebook. Having a target of 50 code vectors, we have to prescribe at each step, as discussed above, typically 50/4.6 = 11 prototypes. As seen in figure 5a, we choose 11 initial prototypes uniformly distributed along the bisecting line. Using the training set of vectors of the figure 2, the induced metastable states are detected with their corresponding bassins of attraction. The 11 most frequent, prescribed or induced, attractors are selected and the 11 centers of gravity of their bassins of attraction are taken as new prototypes (figure 5b ). After 3 iterations, the distorsion measure stabilizes (Table 1).

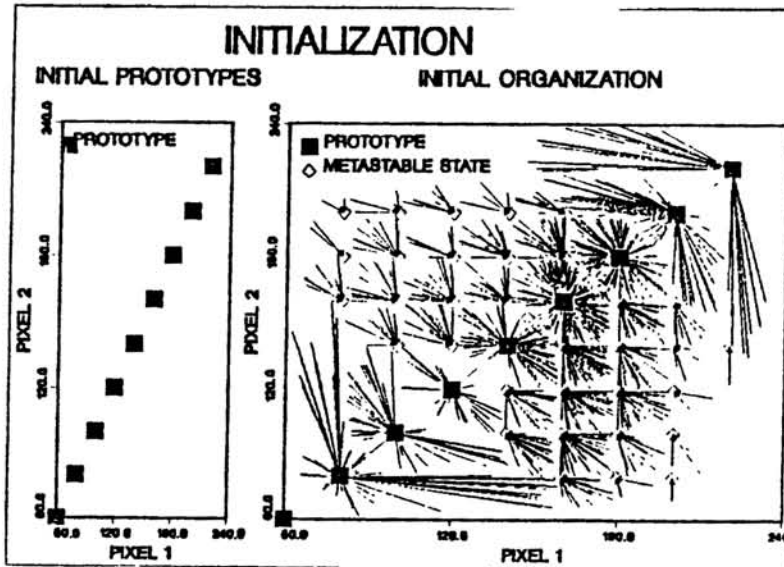

Figure 5a : Initialization of the self-organizing scheme.

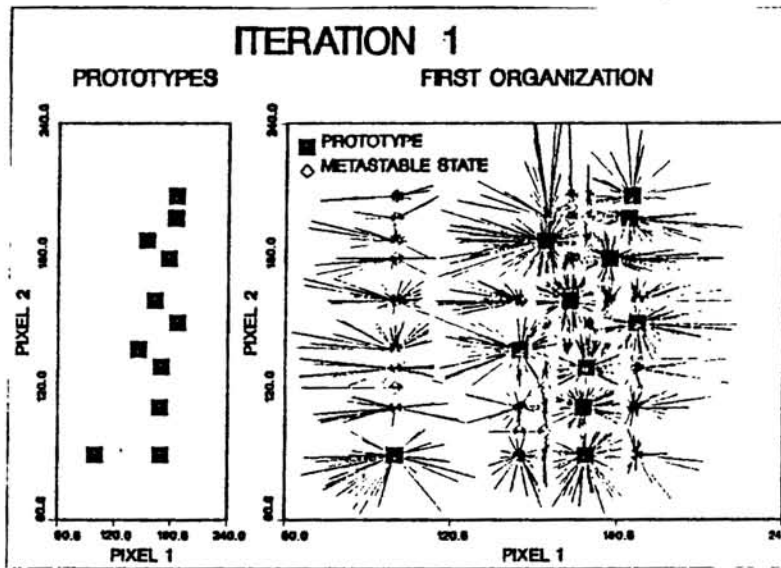

Figure 5b : First iteration of the self-organizing scheme.

| no of iterations | | global distorsion | codebook size | generalized attractors |
|---|---|---|---|---|
| | 1 | 157 | 53 | 0 |
| | 2 | 103 | 57 | 4 |
| | 3 | 97 | 79 | 20 |
| | 4 | 97 | 84 | 20 |
| | 5 | 98 | 68 | 15 |

Table 1 : Evolution of the distorsion measure versus the iterations of the self-organizing scheme. It stabilizes in 3 iterations.

Fourty lines of a TV image (the port of Baltimore) of 8
bits per pel, has been coded with an adaptive neural
codebook of 50 2D-vectors. The coherence of the coding is
visible from the apparent continuity of the image
(Figure 6).
The coded image has 2.5 bits per pel.

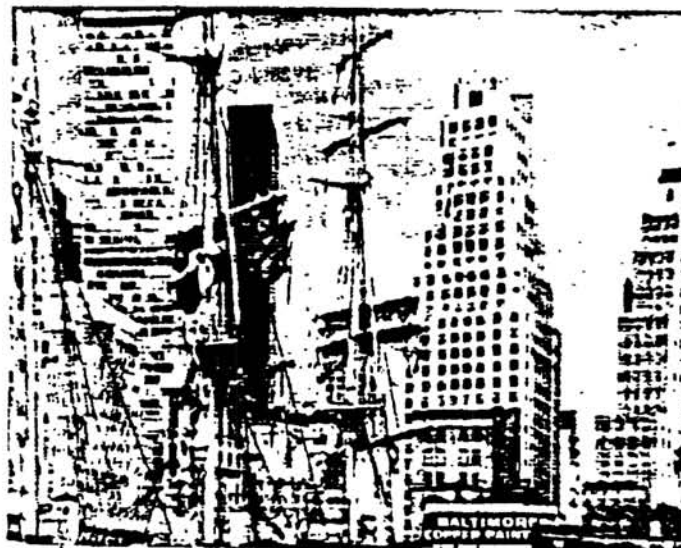

Figure 6 : Neural coded image with 2.5 bits per pel.

## CONCLUSION

Using a "classical" clustering algorithm, a
self-organizing scheme has been developed in a Hopfield
network for the adaptive design of a codebook of small
dimension vectors in a Vector Quantization technique. It
has been shown that using the Minimal Overlap
prescription rule, the metastable states induced in a
spin glass-like network can be used as extra-codes. The
optimal organization of the prescribed and induced
attractors, has been defined as the limit organization
obtained from the iterative learning process. It is an
example of "learning by selection" as already proposed by
physicists and biologists (Toulouse and al. 1986).
Hardware implementation on the neural VLSI circuit
currently designed at LEP should allow for on-line
codebook computations.

We would like to thank J.J. Hopfield who has inspired
this study as well H. Bosma and W. Kreuwels from Philips
Research Laboratories, Eindhoven, who have allow to
initialize this research.

## REFERENCES

1 - J.J. Hopfield, Proc. Nat. Acad. Sci. USA, 79, 2554 - 2558 (1982); J.J. Hopfield and D.W. Tank , Science 233 , 625 (1986) ; W.A. Little, Math. Biosi., 19, 101-120 , (1974).

2 - D.J. Amit, H. Gutfreund, and H. Sompolinsky, Phys.Rev. 32, Ann. Phys. 173, 30 (1987).

3 - L. Personnaz, I. Guyon and G. Dreyfus, J. Phys. Lett. 46, L359 (1985).

4 - J.A. Hertz, 2nd International Conference on Vector and parallel computing, Tromso, Norway, June (1988).

5 - M.A. Virasoro, Disorder Systems and Biological Organization, ed. E. Bienenstock, Springer, Berlin (1985); H. Gutfreund (Racah Institute of Physics, Jerusalem)(1986); C. Cortes, A. Krogh and J.A. Hertz, J. of Phys. A., (1986).

6 - R.M. Gray, IEEE ASSP Magazine 5 (Apr. 1984).

7 - L.D. Jackel , R.E. Howard , J.S. Denker , W. Hubbard and S.A. Solla, Applied Optics, Vol. 26, 23, (1987).

8 - T. Kononen, Finland, Helsinky University of Technology, Tech. Rep. No. TKK-F-A601 ; T. Kohonen, Neural Networks, 1, Number 1, (1988).

9 - S. Grossberg, Cognitive Sci., 11, 23-63 (1987).

10 - G.W. Cottrell , P. Murro and D.Z. Zipser, Institute of cognitive Science, Report 8702 (1987).

11 - D. Sherrington and S. Kirkpatrick, Phys. Rev. Lett. 35, 1792 (1975); G. Toulouse, Commun. Phys. 2, 115-119 (1977); M. Mezard , G. Parisi , N. Sourlas , G. Toulouse and M. Virasoro, Phys. Dev. Lett., 52, 1156-1159 (1984).

12 - W. Krauth and M. Mezard , J. Phys.A : Math. Gen. 20, L745-L752 (1987)

13 - M. Naillon and J.B. Theeten, to be published.

14 - G. Toulouse, S. Dehaene and J.P. Changeux, Pro. Natl.Acad. Sci. USA, 83, 1695, (1986).
